# Kernel Machines and Boolean Functions

**Adam Kowalczyk**
Telstra Research Laboratories
Telstra, Clayton, VIC 3168
a.kowalczyk@trl.oz.au

**Alex J. Smola, Robert C. Williamson**
RSISE, MLG and TelEng
ANU, Canberra, ACT, 0200
{Alex.Smola, Bob.Williamson}@anu.edu.au

## Abstract

We give results about the learnability and required complexity of logical formulae to solve classification problems. These results are obtained by linking propositional logic with kernel machines. In particular we show that decision trees and disjunctive normal forms (DNF) can be represented by the help of a special kernel, linking regularized risk to separation margin. Subsequently we derive a number of lower bounds on the required complexity of logic formulae using properties of algorithms for generation of linear estimators, such as perceptron and maximal perceptron learning.

## 1   Introduction

The question of how many Boolean primitives are needed to learn a logical formula is typically an NP-hard problem, especially when learning from noisy data. Likewise, when dealing with decision trees, the question what depth and complexity of a tree is required to learn a certain mapping has proven to be a difficult task.

We address this issue in the present paper and give *lower* bounds on the number of Boolean functions required to learn a mapping. This is achieved by a constructive algorithm which can be carried out in polynomial time. Our tools for this purpose are a Support Vector learning algorithm and a special polynomial kernel.

In Section 2 we define the classes of functions to be studied. We show that we can treat propositional logic and decision trees within the same framework. Furthermore we will argue that in the limit boosted decision trees correspond to polynomial classifiers built directly on the data. Section 3 contains our main result linking the margin of separation to a simple complexity measure on the class of logical formulae (number of terms and depth). Subsequently we apply this connection to devise test procedures concerning the complexity of logical formulae capable of learning a certain dataset. More specifically, this will involve the training of a perceptron to minimize the regularized risk functional. Experimental results and a discussion conclude the paper. Some proofs have been omitted due to space constraints. They can be found in an extended version of this paper (available at `http://www.kernel-machines.org`).

## 2   Polynomial Representation of Boolean Formulae

We use the standard assumptions of supervised learning: we have a training set $\{(\mathbf{x}_1, y_1), \ldots, (\mathbf{x}_m, y_m)\} \subset \mathbf{X} \times Y$. Based on these observations we attempt to find a function $f : \mathbf{X} \to Y$ which incorporates the information given by the training set. Here goodness of fit is measured relative to some predefined loss function or a probabilistic model.

What makes the situation in this paper special is that we assume that $\mathbf{X} \subseteq \mathbf{B}^n$ where $\mathbf{B} = \{0, 1\}$ and moreover $Y = \{\pm 1\}$. In other words, we attempt to learn a binary function on Boolean variables. A few definitions will be useful.

The set of all polynomials $f$ of degree $\leq d \in \mathbb{N}$ on $\mathbf{B}^n$ will be denoted[1] by $\mathrm{Pol}^d$. They are given by the expansion

$$f(\mathbf{x}) = \sum_{\mathbf{i} \in \mathbf{I}} c_{\mathbf{i}} \mathbf{x}^{\mathbf{i}} \text{ where } \mathbf{I} \subset \{\mathbf{i} | \mathbf{i} \in \mathbf{B}^n \text{ and } |\mathbf{i}| \leq d\}, \tag{1}$$

for every $\mathbf{x} = (x_1, .., x_n) \in X$, where $c_{\mathbf{i}} \in \mathbb{R}$ and we use a compact notation $\mathbf{x}^{\mathbf{i}} := x_1^{i_1} \cdots x_n^{i_n}$ for every $(i_1, ..., i_n) \in \mathbf{B}^n$ for monomials on $\mathbf{B}^n$, with the usual convention of $x^0 := 1$ for every $x \geq 0$. In order to avoid further notation we *always* assume in such expansions that $c_{\mathbf{i}} \neq 0$ for all $\mathbf{i} \in \mathbf{I}$.

The subset $\mathrm{DNF}^d \subset \mathrm{Pol}^d$ of all polynomials of the form

$$f(\mathbf{x}) = -1 + 2 \sum_{\mathbf{i} \in \mathbf{I}} \mathbf{x}^{\mathbf{i}} \text{ where } \mathbf{I} \subset \{\mathbf{i} | \mathbf{i} \in \mathbf{B}^n \text{ and } |\mathbf{i}| \leq d\} \tag{2}$$

will be called *disjunctive normal forms*. It is linked by the following lemma to the set of disjunctive normal forms $\mathrm{DNF}^d_{\mathrm{logic}}$ commonly used in the propositional logic. The latter consist of all clauses $\phi : \mathbf{B}^n \to \{\mathrm{FALSE}, \mathrm{TRUE}\}$ which can be expressed by disjunctions of terms, each being a conjunction of up to $d$ logical primitives $[x_i = 1]$ and $[x_i = 0]$.

**Lemma 1**  *Assume for each $i \in \{1, \ldots, n\}$ there exists an $i' \in \{1, \ldots, n\}$ such that*

$$x_i = 1 - x_{i'} \text{ for all } \mathbf{x} = (x_1, ..., x_n) \in X. \tag{3}$$

*Then for every $f \in \mathrm{DNF}^d$ there exists $\phi \in \mathrm{DNF}^d_{\mathrm{logic}}$ such that for every $\mathbf{x} \in X$*

$$f(\mathbf{x}) \geq 1 \text{ if and only if } \phi(\mathbf{x}) \equiv \mathrm{TRUE}.$$

*And vice versa, for every such $\phi$ there exists $f$ satisfying the above relation.*

**Standing Assumption:** Unless stated otherwise, we will in the rest of the paper assume that (3) of the above lemma holds. This is not a major restriction, since we can always satisfy (3) by artificially enlarging $X \subset \mathbf{B}^n$ into $\{(x_i, 1 - x_i) \; ; \; (x_i) \in X\} \subset \mathbf{B}^{2n}$.

Now we consider another special subclass of polynomials, $\mathrm{DT}^d(X) \subset \mathrm{Pol}^d$, called *decision trees* . These are polynomials which have expansions of type (1) where $(i)$ all coefficients $c_{\mathbf{i}} \in \{-1, 1\}$ and $(ii)$ for every $\mathbf{x} \in X$ exactly one monomial $\mathbf{x}^{\mathbf{i}}$, $\mathbf{i} \in \mathbf{I}$, 'fires', i.e. exactly one of the numbers $\{\mathbf{x}^{\mathbf{i}}\}_{i \in I}$ equals 1 and all the others are 0.

Eq. (1) shows that each decision tree can be expressed as half of a difference between two disjunctive normal forms such that for any given input, one and only one of the conjunctions comprising them will be true. There exists also an obvious link to popular decision trees (on Boolean variables) used for classification in machine learning, cf. [4, 12]. Here the depth of a leaf equals the degree of the corresponding monomial, and the coefficient $c^{\mathbf{i}} \in \{\pm 1\}$ corresponds to the class associated with the leaf.

# 3  Reproducing Kernel Hilbert Space and Risk

**Kernel**   The next step is to map the complexity measure applied to decision trees, such as depth or number of leaves, to a Reproducing Kernel Hilbert Space (RKHS), as used in Support Vector machines. This is defined as $\mathcal{H} = \mathrm{Pol}^d$ with the scalar product corresponding to the norm defined via the quadratic form on $f$

$$\|f\|_{\mathcal{H}}^2 := \sum_{\mathbf{i} \in I} K_{|\mathbf{i}|} c_{\mathbf{i}}^2. \tag{4}$$

Here $K_i > 0$ with $i \in \{0, \dots, d\}$ are *complexity weights* for each degree of the polynomials and the coefficients $c_{\mathbf{i}}$ are the coefficients of expansion (1).

**Lemma 2 (Existence of Kernel)** *The RKHS kernel $k$ realizing the dot product corresponding to the quadratic form* (4) *with $k(\mathbf{x}, \cdot) \in \mathrm{Pol}^d$ has the following efficient functional form:*

$$k(\mathbf{x}, \mathbf{x}') = \sum_{j=0}^{d} K_j^{-1} \binom{\langle \mathbf{x}, \mathbf{x}' \rangle}{j}. \tag{5}$$

**Proof**   The norm $\|f\|_{\mathcal{H}}$ is well defined by (4) for all $f \in \mathrm{Pol}^d$ and the space $\mathrm{Pol}^d$ is complete. Furthermore it is easy to check that (4) defines a homogeneous quadratic form on $\mathrm{Pol}^d$. Via the polarization identity we can reconstruct a bilinear form (dot product) from (4). This gives us the desired Hilbert space. From [1] we obtain that there exists a unique kernel $k(\mathbf{x}, \mathbf{x}')$ corresponding to $\|\cdot\|_{\mathcal{H}}^2$. The key observation for derivation of its form (5) is that given $\mathbf{x}, \mathbf{x}' \in X$ and $j > 0$ there are exactly $\binom{\langle \mathbf{x}, \mathbf{x}' \rangle}{j}$ non-vanishing monomials of the form $\mathbf{x}^{\mathbf{i}} \mathbf{x}'^{\mathbf{i}} = x_{\alpha_1} x'_{\alpha_1} \cdots x_{\alpha_j} x'_{\alpha_j}$, where $1 \leq \alpha_1 < \alpha_2 < \cdots < \alpha_j \leq n$ are positions of 1's in the sequence $\mathbf{i}$. ■

Note that for the special case where $K_j = \epsilon^{-j}$ with $\epsilon > 0$ and $d \geq \langle \mathbf{x}, \mathbf{x}' \rangle$, (5) simply leads to a binomial expansion and we obtain

$$k(\mathbf{x}, \mathbf{x}') = \sum_{j=0}^{d} \epsilon^j \binom{\langle \mathbf{x}, \mathbf{x}' \rangle}{j} = (1 + \epsilon)^{\langle \mathbf{x}, \mathbf{x}' \rangle}. \tag{6}$$

The larger $\epsilon$, the less severely we will penalize higher order polynomials, which provides us with an effective means of controlling the complexity of the estimates. Note that this is applicable to the case when $d \geq |\mathbf{x}|$, and always holds for $d = n$.

Due to the choice of the $c_{\mathbf{i}}$ in $\mathrm{DNF}^d$ and $\mathrm{DT}^d(X)$ we obtain

$$\|f\|_{\mathcal{H}}^2 = 1 + 4 \sum_{\mathbf{i} \in I} K_{|\mathbf{i}|} \text{ for } f \in \mathrm{DNF}^d \text{ and } \|f\|_{\mathcal{H}}^2 = \sum_{\mathbf{i} \in I} K_{|\mathbf{i}|} \text{ for } f \in \mathrm{DT}^d(X).$$

Next we introduce regularized risk functionals. They follow the standard assumptions made in soft-margin SVM and regularization networks.

For our training set $(\mathbf{x}_i, y_i)$ of size $m$ and a regularization constant $\lambda > 0$ we define

$$\mathcal{R}[f, \lambda] := \|f\|_{\mathcal{H}}^2 + \lambda^{-1} \sum_{i=1}^{m} (1 - y_i f(\mathbf{x}_i))^2,$$

$$\mathcal{R}_+[f, \lambda] := \|f\|_{\mathcal{H}}^2 + \lambda^{-1} \sum_{i=1}^{m} [1 - y_i f(\mathbf{x}_i)]_+^2,$$

for every $f \in \mathrm{Pol}^d$, where $[\xi]_+ := \max(0, \xi)$ for every $\xi \in \mathbb{R}$.

The first risk is typically used by regularization networks [8], the other by support vector machines [5]. Note that for all $f \in \mathrm{Pol}^d$ we have $\mathcal{R}[f, \lambda] \geq \mathcal{R}_+[f, \lambda]$. Furthermore, if $f \in \mathrm{DNF}^d \cup \mathrm{DT}^d(X)$, then $|f(\mathbf{x}_i)| \geq 1$ and hence

$$\mathcal{R}[f, \lambda] \geq \mathcal{R}_+[f, \lambda] \geq \sum_{\mathbf{i} \in \mathbf{I}}^I K_{|\mathbf{i}|} + 4\lambda^{-1} \, \mathrm{err}(f) \tag{7}$$

$$\text{where} \qquad \mathrm{err}(f) := \#\{i \mid y_i \neq f(\mathbf{x}_i)\} \tag{8}$$

denotes the number of *classification errors* (on the training set).

Note that in (7) equalities hold throughout for $f \in \mathrm{DT}^d(X)$ and in such a case the risks are fully determined by the depths of the leaves of the decision tree and the number of classification errors. Furthermore, in the particular case of decision trees and all coefficients $K_{|i|} = 1$, i.e. when $\|f\|_K^2$ equals to the number of leaves of the decision tree $f \in \mathrm{DT}^d(X)$, the regularized risks $\mathcal{R}[f, \lambda] = \mathcal{R}_+[f, \lambda]$ are exactly equal to the "cost complexity" employed to prune decision trees by CART algorithm [4]. In other words, the basis of the pruning algorithm in CART is the minimisation of the regularised risk in the class of subtrees of the maximal tree, with the regularisation constant $\lambda$ selected by a heuristic applied to a validation set.

Our reasoning in the following relies on the idea that if we can find some function $f \in \mathrm{Pol}^d$ which minimizes $\mathcal{R}[f, \lambda]$ or $\mathcal{R}_+[f, \lambda]$, then the minimizer of the risk functionals, when chosen from the more restrictive set $f \in \mathrm{DT}^d(X)$ or $f \in \mathrm{DNF}^d$, must have a risk functional at least as large as the one found by optimizing over $\mathrm{Pol}^d$. This can then be translated into a lower bound on the complexity of $f$ since $\mathrm{DT}^d(X), \mathrm{DNF}^d \subset \mathrm{Pol}^d$.

## 4  Complexity Bounds

The last part missing to establish a polynomial-time device to lower-bound the required complexity of a logical formula is to present actual algorithms for minimizing $\mathcal{R}[f, \lambda]$ or $\mathcal{R}_+[f, \lambda]$. In this section we will study two such methods: the kernel perceptron and the maximum margin perceptron and establish bounds on execution time and regularized risk.

**Kernel Perceptron Test** The $k, \lambda$-perceptron learning algorithm is a direct modification of ordinary linear perceptron learning rule. In the particular case of $\lambda = 0$ it becomes the ordinary perceptron learning rule in the feature space $\mathbb{R}^N$. For $\lambda > 0$ it implements perceptron learning rule in the extended feature space $\mathbb{R}^N \times \mathbb{R}^m$; cf. [7, 6] for details.

---

**Algorithm 1** Regularized kernel perceptron ($k, \lambda$-perceptron)

---

Given: a Mercer kernel $k$ and a constant $\lambda \geq 0$.
Initialize: $t = 0$ and $\alpha_i = 0$ for $i = 1, ..., m$.
**while** an update is possible **do**
    find $j$ such that $y_j \sum_{i=1}^m y_i \alpha_i k(\mathbf{x}_j, \mathbf{x}_i) + \lambda \alpha_j \leq 0$, then update:

$$\alpha_j \leftarrow \alpha_j + 1 \text{ and } t \leftarrow t + 1.$$

**end while**

---

We introduce the special notation: $R = \max_i \sqrt{k(\mathbf{x}_i, \mathbf{x}_i)}$, $r = \min_i \sqrt{k(\mathbf{x}_i, \mathbf{x}_i)}$ and $f_{\vec{\alpha}, k}(\mathbf{x}) := \sum_i y_i \alpha_i k(\mathbf{x}_i, \mathbf{x})$ for every $\mathbf{x} \in X$ and $\vec{\alpha} \in \mathbb{R}^m$. Note that $f_{\vec{\alpha}, k} \in \mathrm{Pol}^d$ and $\|f_{\vec{\alpha}, k}\|_{\mathcal{H}}^2 = \sum_{ij} y_i y_j \alpha_i \alpha_j k(\mathbf{x}_i, \mathbf{x}_j)$.

A modification of the standard proof of convergence of linear perceptron [11] combined with the extended feature space trick [13] gives the following result.

**Theorem 3** *Assume that the coefficients $\vec{\alpha} = (\alpha_i) \in \mathbb{R}^m$ were generated after t-th update of the $k, \lambda$-perceptron and*

$$\rho_* := \left[ \max_{\vec{\alpha} \in \mathbb{R}^m - 0} \min_i y_i f_{\vec{\alpha}, k}(\mathbf{x}_i) / \|f_{\vec{\alpha}, k}\|_{\mathcal{H}} \right]_+ .$$

*Then $t \leq (R^2 + \lambda)/({\rho_*}^2 + \lambda/m)$ and*

$$\mathcal{R}[f, \lambda] \geq \mathcal{R}_+[f, \lambda] \geq \frac{t^2}{\|f_{\vec{\alpha}, k}\|_{\mathcal{H}}^2 + \lambda \|\vec{\alpha}\|^2} \geq \frac{t}{R^2 + \lambda} \quad \text{for every } f \in \text{Pol}^d. \tag{9}$$

Note that $\rho_*$ defined above is the maximal margin of separation of the training data by polynomials from $\text{Pol}^d$ (treated as elements of the RKHS).

**Maximum Margin Perceptron Test** Below we state formally the soft margin version of maximal margin perceptron algorithm. This is a simplified (homogeneous) version of the algorithm introduced in [9].

---

**Algorithm 2** Greedy Maximal Margin Perceptron ($k, \lambda$-MMP)

---

Given: $\epsilon > 0$, $\lambda \geq 0$ a and a Mercer kernel $k$.
Initialize: $k^*_j = k(\mathbf{x}_j, \mathbf{x}_j) + \lambda$ for $j = 1, .., m$;
$i = \arg\min_j k^*_j$, $\|f\|^2 = k^*_i$, $\tau_i = 0$;
$f_j = y_j k(\mathbf{x}_i, \mathbf{x}_j) + \delta_{ij}\lambda$ and $\alpha_j = \delta_{ij}$ for $j = 1, ..., m$;
**while** $\exists_{i'} y_{i'} \mathbf{w} \cdot \mathbf{x}_{i'} \leq (1 - \epsilon)\|f\|^2$ **do**
  **for** for every $j = 1, .., m$ **do**
    $f_j \leftarrow (1 - \tau_i)f_j + \tau_i y_i y_j k(\mathbf{x}_i, \mathbf{x}_j) + \lambda \tau_i \delta_{ij}$;
    $g_j \leftarrow \|f\|^2 - f_j$; $d_j = \|f\|^2 + k^*_j - 2f_j$;
    $\alpha_j \leftarrow \tau_i \delta_{ij} + (1 - \tau_i)\alpha_j$;
    $\tau_j \leftarrow \frac{g_j}{d_j}$ if $g_j > 0$, else $\leftarrow \max(\frac{-\alpha_j}{1-\alpha_j}, \frac{g_j}{d_j})$ if $g_j < 0$ & $0 < \alpha_j < 1$, else $\leftarrow 0$.
  **end for**
  find $i = \arg\max_j(\tau_j g_j)$, then set
  $\|f\|^2 \leftarrow (1 - \tau_i)^2\|f\|^2 + \tau_i k^*_i + 2\tau_i(1 - \tau_i)f_i$ ;
**end while**

---

The proof of the following theorem uses the extended feature space [13].

**Theorem 4** *Given $0 < \epsilon$ and $\lambda \geq 0$. Assume that the vector $\vec{\alpha} = (\alpha_i) \in \mathbb{R}^m$ was generated after t-th iteration of the "while loop" of the $k, \lambda$-MMP learning rule. Then*

$$t \quad \leq \quad \frac{R^2 + r^2 + 2\lambda}{\epsilon^2} \left( \frac{1}{{\rho_*}^2 + \lambda/m} - \frac{1}{r^2 + \lambda} \right) \tag{10}$$

$$\mathcal{R}[f, \lambda] \quad \geq \quad \mathcal{R}_+[f, \lambda] \geq \frac{1}{\|f_{\vec{\alpha}, k}\|_{\mathcal{H}}^2 + \lambda \|\vec{\alpha}\|^2} \geq \frac{1}{r^2 + \lambda} + \frac{t\epsilon^2}{R^2 + r^2 + 2\lambda} \tag{11}$$

*for every $f \in \text{Pol}^d$. If the algorithm halts after t-th update, then*

$$\frac{1}{\|f_{\vec{\alpha}, k}\|_{\mathcal{H}}^2 + \lambda \|\vec{\alpha}\|^2} \geq \min_{f \in \text{Pol}^d} \mathcal{R}_+[f, \lambda](1 - \epsilon)^2. \tag{12}$$

Note that condition (10) ensures the convergence of the algorithm in a finite time. The above theorem for $\lambda = 0$ ensures that solution generated by Algorithm 2 converges to the (hard) maximum margin classifier. Further, it can be shown that the bound (11) holds for every $\vec{\alpha} = (\alpha_i)$ such that each $\alpha_i \geq 0$ and $\sum \alpha_i = 1$.

**Bounds on classification error** The task of finding a linear perceptron minimizing the number of classification errors on the training set is known to be NP-hard. On this basis it is reasonable to expect that finding a decision tree or disjunctive normal form of upper bounded complexity and minimizing the number of errors is also hard. In this section we provide a lower bound on the number of errors for such classifiers.

The following estimates on $\mathrm{err}(f)$, i.e. the number of classification errors (8), can be derived from Theorems 3 and 4:

**Theorem 5** *Let* $\lambda, \epsilon > 0$ *and* $f \in \mathrm{DT}^d$. *If the vector* $\vec{\alpha} = (\alpha_i) \in \mathbb{R}^m$ *has been generated after t-th iteration of the "while loop" of the* $k, \lambda$*-perceptron learning rule, then*

$$\mathrm{err}(f) \geq \frac{\lambda}{4} \left( \frac{t^2}{||f_{\vec{\alpha},k}||^2_{\mathcal{H}} + \lambda ||\vec{\alpha}||^2} - ||f||^2_{\mathcal{H}} \right) \geq \frac{\lambda}{4} \left( \frac{t\epsilon^2}{R^2 + \lambda} - ||f||^2_{\mathcal{H}} \right). \qquad (13)$$

*On the other hand, if* $\vec{\alpha} = (\alpha_i) \in \mathbb{R}^m$ *has been generated after t-th iteration of the "while loop" of the* $k, \lambda$*-MMP learning rule, then*

$$\mathrm{err}(f) \quad \geq \quad \frac{\lambda}{4} \left( \frac{1}{||f_{\vec{\alpha},k}||^2_{\mathcal{H}} + \lambda ||\vec{\alpha}||^2} - ||f||^2_{\mathcal{H}} \right) \qquad (14)$$

$$\geq \quad \frac{\lambda}{4} \left( \frac{1}{r^2 + \lambda} + \frac{t\epsilon^2}{R^2 + r^2 + 2\lambda} - ||f||^2_{\mathcal{H}} \right). \qquad (15)$$

*Additionally, the estimate* (14) *holds for every* $\vec{\alpha} = (\alpha_i) \in \mathbb{R}^m$ *such that* $\sum \alpha_i = 1$ *and each* $\alpha_i \geq 0$.

Note that $\sum \alpha_i$ equals $t$ in (13), while it is 1 in (14). The following result is derived form some recent results of Ben David and Simon [2] on efficient learning of perceptrons.

**Theorem 6** *Given* $\mu > 0$ *and integer* $d > 0$. *There exists an algorithm* $A_\mu$ *which runs in time polynomial in both the input dimension* $n$ *and the number of training samples* $m$, *that given the labelled training sample* $(\mathbf{x}_i, y_i)$, $i = 1, ..., m$, *it outputs a polynomial* $h \in \mathrm{Pol}^d$ *such that* $\mathrm{err}(h) \leq \mathrm{err}(f)$, *for every in* $f \in \mathrm{DT}^d \cup \mathrm{DNF}^d$.

Following [2] we give an explicit formulation of the algorithm $A_\mu$: for each subset of $\leq \lceil 4/\mu^4 \rceil$ elements of the training set $\{(\mathbf{z}_i, y_i)\}_{i=1,...,m}$ find the maximal margin hyperplane, if one exists. Using the standard quadratic programming approach this can be done in time polynomial in both $N$ and $m$ [3]. Next, define $\mathbf{w}_h \in \mathbb{R}^N$ as the vector of the hyperplane with the lowest error rate on the whole training set. Finally, set $h(\cdot) := \mathbf{w}_h \cdot \Phi(\cdot) \in \mathrm{Pol}^d$.

## 5 Experimental Results and Discussion

We have used a standard machine learning benchmark of noisy 7 bit LED display for 10 digits, 0 though 9, originally introduced in [4]. We generated 500 examples for training and 5000 for independent test, under assumption of 10% probability of a bit being reversed. The task set was to discriminate between two classes, digits 0-4 and digits 5-9. Each "noisy digit" data vector $(x_1, ...., x_7)$ was complemented by an additional 7 bits vector $(1 - x_1, ..., 1 - x_7)$ to ensure that our Standing Assumption of Section 2 holds true.

For a sake of simplicity we used fixed complexity weights, $K_i = 1, i = 0, ..., d$, and $\lambda = 4$, which for a decision tree $f \in \mathrm{DT}^d$ gives a simple formula for the risk

$$\mathcal{R}_+[f, \lambda] = \mathcal{R}[f, \lambda] = [\text{number of leaves}] + [\text{number of errors}].$$

Four different algorithms have been applied to this data: $(i)$ Decision Trees, version C4.5 [12] (available from www.cse.unsw.edu.au/∼quinlan/), $(ii)$ regularized kernel perceptron (Algorithm 1) with the generated coefficients scaled $\vec{\alpha} \rightarrow \vec{\alpha}/(t * (\|f_{\vec{\alpha},k}\|_{\mathcal{H}} + \lambda \|\vec{\alpha}\|^2))$, where $t$ is the number of updates to the convergence, $(iii)$ greedy maximal margin classifier (Algorithm 2) and $(iv)$ mask perceptron [10] which for this data generates a polynomial $f \in \mathrm{Pol}^d$ using some greedy search heuristics. Table 1 gives the experimental results.

Table 1: Results for recognition of two groups of digits on faulty LED-display.

| Algorithm | Risk (no. of leaves /SV/terms ) | | Error rate %: train/test | |
|---|---|---|---|---|
| | $d = 3$ | $d = 7$ | $d = 3$ | $d = 7$ |
| Decision tree | 110 (4 leaves) | 80 (17 leaves) | 21.3 / 22.9 | 12.0 / 15.8 |
| Kernel SVM | 44.4(413 SV) | 40.8 (382 SV) | 12.2 / 15.1 | 11.2 / 14.8 |
| Kernel percep. | 53.1 (294 SV) | 54.9 (286 SV) | 11.8 / 16.3 | 13.8 / 17.1 |
| Mask percep. | 53.2(10 terms) | 49.1 (26 terms) | 12.8 / 15.7 | 11.8 / 15.6 |

The lower bound on risk from maximal margin criterion (Eq. 11) are 44.3 and 40.7 for $d = 3$ and $d = 7$, respectively. Similarly, the lower bound on risk from kernel perceptron criterion (Eq. 9) were 39.7 and 36.2, respectively. Risks for SVM solutions approach this bound and for kernel perceptron they are reasonably close. Comparison with the risks obtained for decision trees show that our lower bounds are meaningful (for the "un-pruned" decision trees risks were only slightly worse). The mask perceptron results show that simple (low number of terms) polynomial solutions with risks approaching our lower bounds can be practically found.

The Bayes-optimal classifier can be evaluated on this data set, since we know explicitly the distribution from which data is drawn. Its error rates are 11.2% and 13.8% on the training and test sets, respectively. SVM solutions have error rates closest to the Bayesian classifier (the test error rate for $d = 7$ exceeds the one of the Bayes-optimal classifier by only 7%).

**Boosted Decision Trees**   An obvious question to ask is what happens if we take a large enough linear combination of decision trees. This is the case, for instance, in boosting. We can show that $\mathrm{Pol}^d$ is spanned by $\mathrm{DT}^d(X)$. In a nutshell, the proof relies on the partition of the identity into

$$1 = \sum_{\mathbf{a+b=i}} \mathbf{x^a \bar{x}^b} \text{ where } \bar{\mathbf{x}} = (1 - x_1, 1 - x_2, \dots, 1 - x_n)$$

and solving this expansion for $\mathbf{x^i}$, where the remainder turns out to be a decision tree. This means that in the limit, boosting decision trees finds a maximum margin solution in $\mathrm{Pol}^d$, a goal more directly achievable via a maximum margin perceptron on $\mathrm{Pol}^d$.

**Conclusion**   We have shown that kernel methods with their analytical tools are applicable well outside of their traditional domain, namely in the area of propositional logic, which traditionally has been an area of discrete, combinatorial rather then continuous analytical methods. The constructive lower bounds we proved offer a fresh approach to some seemingly intractable problems. For instance, such bounds can be used as points of reference for practical applications of inductive techniques like as decision trees.

The use of Boolean kernels introduced here allows a more insightful comparison of performance of logic based and analytical, linear machine learning algorithms.

This contributes to the research in the theory of learning systems as illustrated by the result on existence of polynomial time algorithm for estimation of minimal number of training errors for decision trees and disjunctive normal forms.

A potentially more practical link, to boosted decision trees, and their convergence to the maximum margin solutions has to be investigated further. The current paper sets foundations for such research.

Boolean kernels can potentially stimulate more accurate (kernel) support vector machines by providing more intuitive construction of kernels. This is the subject of ongoing research.

**Acknowledgments**    A.K. acknowledges permission of the Chief Technology Officer, Telstra to publish this paper. A.S. was supported by a grant of the DFG Sm 62/1-1. Parts of this work were supported by the ARC and an R& D grant from Telstra. Thanks to P. Sember and H. Ferra for help in preparation of this paper.

## Footnotes

[1]Such binary polynomials are widely used under the name of score tables, e.g. typically loan applications are assessed by financial institutions by an evaluation of such score tables.

# References

[1] N. Aronszajn. Theory of reproducing kernels. *Transactions of the American Mathematical Society*, 68:337 – 404, 1950.

[2] S. Ben-David and H. U. Simon. Efficient learning of linear perceptron. In T.K. Leen, T.G. Dieterich, and V. Tresp, editors, *Advances in Neural Information Processing Systems 13*, pages 189–195, Cambridge, MA, 2001. MIT Press.

[3] D. P. Bertsekas. *Nonlinear Programming*. Athena Scientific, Belmont, MA, 1995.

[4] L. Breiman, J.H. Friedman, R.A. Olshen, and C.J. Stone. *Classification and Regression Trees*. Wadsworth Int., Belmont, Ca., 1984.

[5] C. Cortes and V. Vapnik. Support vector networks. *Machine Learning*, 20:273 – 297, 1995.

[6] N. Cristianini and J. Shawe-Taylor. *An Introduction to Support Vector Machines and other kernel-based learning methods*. Cambridge University Press, Cambridge, 2000.

[7] Y. Freund and R. E. Schapire. Large margin classification using the perceptron algorithm. In J. Shavlik, editor, *Machine Learning: Proceedings of the Fifteenth International Conference*, San Francisco, CA, 1998. Morgan Kaufmann.

[8] F. Girosi, M. Jones, and T. Poggio. Regularization theory and neural networks architectures. *Neural Computation*, 7(2):219–269, 1995.

[9] A. Kowalczyk. Maximal margin perceptron. In A. Smola, P.Bartlett, B. Schölkopf, and D. Schuurmans, editors, *Advances in Large Margin Classifiers*, pages 61–100, Cambridge, MA, 2000. MIT Press.

[10] A. Kowalczyk and H. Ferrà. Developing higher-order networks with empirically selected units. *IEEE Transactions on Neural Networks*, 5:698–711, 1994.

[11] A. B. Novikoff. On convergence proofs on perceptrons. *Symposium on the Mathematical Theory of Automata*, 12:615–622, 1962.

[12] J.R. Quinlan. Simplifying decision trees. *Int. J. Man-Machine Studies*, **27**:221–234, (1987).

[13] J. Shawe-Taylor and N. Christianini. Margin distribution and soft margin. In A. J. Smola, P. L. Bartlett, B. Schölkopf, and D. Schuurmans, editors, *Advances in Large Margin Classifiers*, pages 349–358, Cambridge, MA, 2000. MIT Press.
